# Identifying Alzheimer's Disease-Related Brain Regions from Multi-Modality Neuroimaging Data using Sparse Composite Linear Discrimination Analysis

**Shuai Huang[1], Jing Li[1], Jieping Ye[2,3], Kewei Chen[4], Teresa Wu[1], Adam Fleisher[4], Eric Reiman[4]**

[1]Industrial Engineering, [2]Computer Science and Engineering, and [3]Center for Evolutionary Medicine and Informatics, The Biodesign Institute, Arizona State University, Tempe, USA
*{shuang31, jing.li.8, jieping.ye, teresa.wu}@asu.edu*
[4]Banner Alzheimer's Institute and Banner PET Center, Banner Good Samaritan Medical Center, Phoenix, USA
*{kewei.chen, adam.fleisher, eric.reiman}@bannerhealth.com*

## Abstract

Diagnosis of Alzheimer's disease (AD) at the early stage of the disease development is of great clinical importance. Current clinical assessment that relies primarily on cognitive measures proves low sensitivity and specificity. The fast growing neuroimaging techniques hold great promise. Research so far has focused on single neuroimaging modality. However, as different modalities provide complementary measures for the same disease pathology, fusion of multi-modality data may increase the statistical power in identification of disease-related brain regions. This is especially true for early AD, at which stage the disease-related regions are most likely to be weak-effect regions that are difficult to be detected from a single modality alone. We propose a sparse composite linear discriminant analysis model (SCLDA) for identification of disease-related brain regions of early AD from multi-modality data. SCLDA uses a novel formulation that decomposes each LDA parameter into a product of a common parameter shared by all the modalities and a parameter specific to each modality, which enables joint analysis of all the modalities and borrowing strength from one another. We prove that this formulation is equivalent to a penalized likelihood with non-convex regularization, which can be solved by the DC (difference of convex functions) programming. We show that in using the DC programming, the property of the non-convex regularization in terms of preserving weak-effect features can be nicely revealed. We perform extensive simulations to show that SCLDA outperforms existing competing algorithms on feature selection, especially on the ability for identifying weak-effect features. We apply SCLDA to the Magnetic Resonance Imaging (MRI) and Positron Emission Tomography (PET) images of 49 AD patients and 67 normal controls (NC). Our study identifies disease-related brain regions consistent with findings in the AD literature.

## 1    Introduction

Alzheimer's disease (AD) is a fatal, neurodegenerative disorder that currently affects over five million people in the U.S. It leads to substantial, progressive neuron damage that is irreversible, which eventually causes death. Early diagnosis of AD is of great clinical importance, because disease-modifying therapies given to patients at the early stage of their disease development will have a much better effect in slowing down the disease progression and helping preserve some cognitive functions of the brain. However, current clinical assessment that majorly relies on cognitive measures proves low sensitivity and specificity in early diagnosis of AD. This is because these cognitive measures are vulnerable to the confounding effect from some non-AD related factors such as patients' mood, and presence of other illnesses or major life events [1]. The confounding effect is especially severe in the diagnosis of early AD, at which time cognitive

impairment is not yet apparent. On the other hand, fast growing neuroimaging techniques, such as Magnetic Resonance Imaging (MRI) and Positron Emission Tomography (PET), provide great opportunities for improving early diagnosis of AD, due to their ability for overcoming the limitations of conventional cognitive measures. There are two major categories of neuroimaging techniques, i.e., functional and structure neuroimaging. MRI is a typical structural neuroimaging technique, which allows for visualization of brain anatomy. PET is a typical functional neuroimaging technique, which measures the cerebral metabolic rate for glucose. Both techniques have been extensively applied to AD studies. For example, studies based on MRI have consistently revealed brain atrophy that involves the hippocampus and entorhinal cortex [2-6]; studies based on PET have revealed functional abnormality that involves the posterior temporal and parietal association cortices [8-10], posterior cingulate, precuneus, and medial temporal cortices [11-14].

There is overlap between the disease-related brain regions detected by MRI and those by PET, such as regions in the hippocampus area and the mesia temporal lobe [15-17]. This is not surprising since MRI and PET are two complementary measures for the same disease pathology, i.e., it starts mainly in the hippocampus and entorhinal cortex, and subsequently spreads throughout temporal and orbiogrontal cortex, poseterior cingulated, and association cortex [7]. However, most existing studies only exploited structural and functional alterations in separation, which ignore the potential interaction between them. The fusion of MRI and PET imaging modalities will increase the statistical power in identification of disease-related brain regions, especially for early AD, at which stage the disease-related regions are most likely to be weak-effect regions that are difficult to be detected from MRI or PET alone. Once a good set of disease-related brain regions is identified, they can be further used to build an effective classifier (i.e., a biomarker from the clinical perspective) to enable AD diagnose with high sensitivity and specificity.

The idea of multi-modality data fusion in the research of neurodegenerative disorders has been exploited before. For example, a number of models have been proposed to combine electroencephalography (EEG) and functional MRI (fMRI), including parallel EEG-fMRI independent component analysis [18]-[19], EEG-informed fMRI analysis [18] [20], and variational Bayesian methods [18] [21]. The purpose of these studies is different from ours, i.e., they aim to combine EEG, which has high temporal resolution but low spatial resolution, and fMRI, which has low temporal resolution but high spatial resolution, so as to obtain an accurate picture for the whole brain with both high spatial and high temporal resolutions [18]-[21]. Also, there have been some studies that include both MRI and PET data for classification [15], [22]-[25]. However, these studies do not make use of the fact that MRI and PET measure the same underlying disease pathology from two complementary perspectives (i.e., structural and functional perspectives), so that the analysis of one imaging modality can borrow strength from the other.

In this paper, we focus on the problem of identifying disease-related brain regions from multi-modality data. This is actually a variable selection problem. Because MRI and PET data are high-dimensional, regularization techniques are needed for effective variable selection, such as the L1-regularization technique [25]-[30] and the L2/L1-regularization technique [31]. In particular, L2/L1-regularization has been used for variable selection jointly on multiple related datasets, also known as multitask feature selection [31], which has a similar nature to our problem. Note that both L1- and L2/L1-regularizations are convex regularizations, which have gained them popularity in the literature. On the other hand, there is increasing evidence that these convex regularizations tend to produce too severely shrunken parameter estimates. Therefore, these convex regularizations could lead to miss-identification of the weak-effect disease-related brain regions, which unfortunately make up a large portion of the disease-related brain regions especially in early AD. Also, convex regularizations tend to select many irrelevant variables to compensate for the overly severe shrinkage in the parameters of the relevant variables. Considering these limitations of convex regularizations, we study non-convex regularizations [33]-[35] [39], which have the advantage of producing mildly or slightly shrunken parameter estimates so as to be able to preserve weak-effect disease-related brain regions and the advantage of avoiding selecting many disease-irrelevant regions.

Specifically in this paper, we propose a sparse composite linear discriminant analysis model, called SCLDA, for identification of disease-related brain regions from multi-modality data. The contributions of our paper include:

- **Formulation:** We propose a novel formulation that decomposes each LDA parameter into a product of a common parameter shared by all the data sources and a parameter specific to each data source, which enables joint analysis of all the data sources and borrowing strength from one another. We further prove that this formulation is equivalent to a penalized likelihood with non-convex regularization.
- **Algorithm**: We show that the proposed non-convex optimization can be solved by the DC (difference of convex functions) programming [39]. More importantly, we show that in using the DC programming, the property of the non-convex regularization in terms of preserving weak-effect features can be nicely revealed.
- **Application**: We apply the proposed SCLDA to the PET and MRI data of early AD patients and normal controls (NC). Our study identifies disease-related brain regions that are consistent with the findings in the AD literature. AD vs. NC classification based on these identified regions achieves high accuracy, which makes the proposed method a useful tool for clinical diagnosis of early AD. In contrast, the convex-regularization based multitask feature selection method [31] identifies more irrelevant brain regions and yields a lower classification accuracy.

## 2    Review of LDA and its variants

Denote $\mathbf{Z} = \{Z_1, Z_2, \dots, Z_p\}^T$ as the variables and assume there are $J$ classes. Denote $N_j$ as the sample size of class $j$ and $N = \sum_{j=1}^J N_j$ is the total sample size. Let $\mathbf{z} = \{\mathbf{z}_1, \mathbf{z}_2, \dots, \mathbf{z}_N\}^T$ be the $N \times p$ sample matrix, where $\mathbf{z}_i$ is the $i^{th}$ sample and $g(i)$ is its associated class index. Let $\hat{\boldsymbol{\mu}}_j = \frac{1}{N_j} \sum_{i=1, g(i)=j}^N \mathbf{z}_i$ be the sample mean of class $j$, $\hat{\boldsymbol{\mu}} = \frac{1}{N} \sum_{i=1}^N \mathbf{z}_i$ be the overall sample mean, $\mathbf{T} = \frac{1}{N} \sum_{i=1}^N (\mathbf{z}_i - \hat{\boldsymbol{\mu}})(\mathbf{z}_i - \hat{\boldsymbol{\mu}})^T$ be the total normalized sum of squares and products (SSQP), $\mathbf{W}_j = \frac{1}{N_j} \sum_{i=1, g(i)=j}^N (\mathbf{z}_i - \hat{\boldsymbol{\mu}}_j)(\mathbf{z}_i - \hat{\boldsymbol{\mu}}_j)^T$ be the normalized class SSQP of class $j$, and $\mathbf{W} = \frac{1}{N} \sum_{j=1}^J N_j \mathbf{W}_j$ be the overall normalized class SSQP.

The objective of LDA is to seek for a $p \times q$ linear transformation matrix, $\boldsymbol{\theta}_q$, with which $\boldsymbol{\theta}_q^T Z$ retains the maximum amount of class discrimination information in $Z$. To achieve this objective, one approach is to seek for the $\boldsymbol{\theta}_q$ that maximizes the between-class variance of $\boldsymbol{\theta}_q^T Z$, which can be measured by $\text{tr}(\boldsymbol{\theta}_q^T \mathbf{T} \boldsymbol{\theta}_q)$, while minimizing the within-class variance of $\boldsymbol{\theta}_q^T Z$, which can be measured by $\text{tr}(\boldsymbol{\theta}_q^T \mathbf{W} \boldsymbol{\theta}_q)$. Here tr() is the matrix trace operator. This is equivalent to solving the following optimization problem:

$$\hat{\boldsymbol{\theta}}_q = \text{argmax}_{\boldsymbol{\theta}_q} \frac{\text{tr}(\boldsymbol{\theta}_q^T \mathbf{T} \boldsymbol{\theta}_q)}{\text{tr}(\boldsymbol{\theta}_q^T \mathbf{W} \boldsymbol{\theta}_q)}. \tag{1}$$

Note that $\hat{\boldsymbol{\theta}}_q$ corresponds to the right eigenvector of $\mathbf{W}^{-1}\mathbf{T}$ and $q = J - 1$.

Another approach used for finding the $\boldsymbol{\theta}_q$ is to use the maximum likelihood estimation for Gaussian populations that have different means and a common covariance matrix. Specifically, as in [36], this approach is developed by assuming the class distributions are Gaussian with a common covariance matrix, and their mean differences lie in a $q$-dimensional subspace of the $p$-dimensional original variable space. Hastie [37] further generalized this approach by assuming that class distributions are a mixture of Gaussians, which has more flexibility than LDA. However, both approaches assume a common covariance matrix for all the classes, which is too strict in many practical applications, especially in high-dimensional problems where the covariance matrices of different classes tend to be different. Consequently, the linear transformation explored by LDA may not be effective.

In [38], a heterogeneous LDA (HLDA) is developed to relax this assumption. The HLDA seeks for a $p \times p$ linear transformation matrix, $\boldsymbol{\theta}$, in which only the first $q$ columns ($\boldsymbol{\theta}_q$) contain discrimination information and the remaining $p - q$ columns ($\boldsymbol{\theta}_{p-q}$) contain no discrimination information. For Gaussian models, assuming lack of discrimination information is equivalent to assuming that the means and the covariance matrices of the class distributions are the same for all

classes, in the $p - q$ dimensional subspace. Following this, the log-likelihood function of $\boldsymbol{\theta}$ can be written as below [38]:

$$l(\boldsymbol{\theta}|\mathbf{Z}) = -\frac{N}{2}\log\left|\boldsymbol{\theta}_{p-q}^T \mathbf{T}\boldsymbol{\theta}_{p-q}\right| - \sum_{j=1}^J \frac{N_j}{2}\log\left|\boldsymbol{\theta}_q^T \mathbf{W}_j \boldsymbol{\theta}_q\right| + N\log|\boldsymbol{\theta}|, \tag{2}$$

Here $|\mathbf{A}|$ denotes the determinant of matrix $\mathbf{A}$. There is no closed-form solution for $\boldsymbol{\theta}$. As a result, numeric methods are needed to derive the maximum likelihood estimate for $\boldsymbol{\theta}$. It is worth mentioning that the LDA in the form of (1) is a special case of the HLDA [38].

## 3    The proposed SCLDA

Suppose that there are multiple data sources, $\{\mathbf{Z}^{(1)}, \mathbf{Z}^{(2)}, \dots, \mathbf{Z}^{(M)}\}$, with each data source capturing one aspect of the same set of physical variables, e.g., the MRI and PET capture the structural and functional aspects of the same brain regions. For each data source, $\mathbf{Z}^{(m)}$, there is a linear transformation matrix $\boldsymbol{\theta}^{(m)}$, which retains the maximum amount of class discrimination information in $\mathbf{Z}^{(m)}$. A naive way for estimating $\boldsymbol{\Theta} = \{\boldsymbol{\theta}^{(1)}, \boldsymbol{\theta}^{(2)}, \dots, \boldsymbol{\theta}^{(M)}\}$ is to separately estimate each $\boldsymbol{\theta}^{(m)}$ based on $\mathbf{Z}^{(m)}$. Apparently, this approach does not take advantage of the fact that all the data sources measure the same physical process. Also, when the sample size of each data source is small, this approach may lead to unreliable estimates for the $\boldsymbol{\theta}^{(m)}$'s.

To tackle these problems, we propose a composite parameterization following the line as [40]. Specifically, let $\theta_{k,l}^{(m)}$ be the element at the $k$-th row and $l$-th column of $\boldsymbol{\theta}^{(m)}$. We treat $\left\{\theta_{k,l}^{(1)}, \theta_{k,l}^{(2)}, \dots, \theta_{k,l}^{(M)}\right\}$ as an interrelated group and parameterize each $\theta_{k,l}^{(m)}$ as $\theta_{k,l}^{(m)} = \delta_k \gamma_{k,l}^{(m)}$, for $1 \le k \le p, 1 \le l \le p$ and $1 \le m \le M$. In order to assure identifiability, we restrict each $\delta_k \ge 0$. Here, $\delta_k$ represents the common information shared by all the data sources about variable $k$, while $\gamma_{k,l}^{(m)}$ represents the specific information only captured by the $m^{th}$ data source. For example, for disease-related brain region identification, if $\delta_k = 0$, it means that all the data sources indicate variable $k$ is not a disease-related brain region; otherwise, variable $k$ is a disease-related brain region. $\gamma_{k,l}^{(m)} \ne 0$ means that the $m^{th}$ data source supports this assertion.

The log-likelihood function of $\boldsymbol{\Theta}$ is:

$$l_0\big(\boldsymbol{\Theta}|\{\mathbf{Z}^{(1)}, \mathbf{Z}^{(2)}, \dots, \mathbf{Z}^{(M)}\}\big) = \sum_{m=1}^M \Bigg\{ -\frac{N^{(m)}}{2}\log\left|\boldsymbol{\theta}_{p-q}^{(m)\,T} \mathbf{T}^{(m)} \boldsymbol{\theta}_{p-q}^{(m)}\right| - \sum_{j=1}^J \frac{N_j^{(m)}}{2}\log\left|\boldsymbol{\theta}_q^{(m)} \mathbf{W}_j^{(m)} \boldsymbol{\theta}_q^{(m)}\right| +$$
$$N^{(m)}\log|\boldsymbol{\theta}^{(m)}| \Bigg\},$$

which follows the same line as (2). However, our formulation includes the following constraints on $\boldsymbol{\Theta}$:

$$\theta_{k,l}^{(m)} = \delta_k \gamma_{k,l}^{(m)}, \delta_k \ge 0, 1 \le k, l \le p, 1 \le m \le M. \tag{3}$$

Let $\boldsymbol{\Gamma} = \left\{\gamma_{k,l}^{(m)}, 1 \le k \le p, \ 1 \le l \le p, 1 \le m \le M\right\}$ and $\boldsymbol{\Psi} = \{\delta_k, 1 \le k \le p\}$. An intuitive choice for estimation of $\boldsymbol{\Gamma}$ and $\boldsymbol{\Psi}$ is to maximize the $l_0\big(\boldsymbol{\Theta}|\{\mathbf{Z}^{(1)}, \mathbf{Z}^{(2)}, \dots, \mathbf{Z}^{(M)}\}\big)$ subject to the constraints in (3). However, it can be anticipated that no element in the estimated $\boldsymbol{\Gamma}$ and $\boldsymbol{\Psi}$ will be exactly zero, resulting in a model which is not interpretable, i.e., poor identification of disease-related regions. Thus, we encourage the estimation of $\boldsymbol{\Psi}$ and the first $q$ columns of $\boldsymbol{\Gamma}$ (i.e., the columns containing discrimination information) to be sparse, by imposing the L1-penalty on $\boldsymbol{\Gamma}$ and $\boldsymbol{\Psi}$. By doing so, we obtain the following optimization problem for the proposed SCLDA:

$$\widehat{\boldsymbol{\Theta}} = \text{argmin}_{\boldsymbol{\Theta}}\, l_1\big(\boldsymbol{\Theta}|\{\mathbf{Z}^{(1)}, \mathbf{Z}^{(2)}, \dots, \mathbf{Z}^{(M)}\}\big) = \text{argmin}_{\boldsymbol{\Theta}} \Big\{ -l_0\big(\boldsymbol{\Theta}|\{\mathbf{Z}^{(1)}, \mathbf{Z}^{(2)}, \dots, \mathbf{Z}^{(M)}\}\big) + \lambda_1 \sum_k \delta_k +$$
$$\lambda_2 \sum_{k,l,m} \gamma_{k,l}^{(m)} \Big\}, \text{subject to}$$
$$\theta_{k,l}^{(m)} = \delta_k \gamma_{k,l}^{(m)}, \delta_k \ge 0, 1 \le k, l \le p, 1 \le m \le M. \tag{4}$$

Here, $\lambda_1$ and $\lambda_2$ control the degrees of sparsity of $\boldsymbol{\Psi}$ and $\boldsymbol{\Gamma}$, respectively. Tuning of two regularization parameters is difficult. Fortunately, we prove the following Theorem which indicates that formulation (4) is equivalent to a simpler optimization problem involving only one regularization parameter.

**Theorem 1**: The optimization problem (4) is equivalent to the following optimization problem:

$$\widetilde{\Theta} = \text{argmin}_{\Theta} \, l_2\big(\Theta|\{\mathbf{Z}^{(1)}, \mathbf{Z}^{(2)}, \dots, \mathbf{Z}^{(M)}\}\big)$$

$$= \text{argmin}_{\Theta} \left\{ -l_0\big(\Theta|\{\mathbf{Z}^{(1)}, \mathbf{Z}^{(2)}, \dots, \mathbf{Z}^{(M)}\}\big) + \lambda \sum_k \sqrt{\sum_{l=1}^{q} \sum_{m=1}^{M} |\theta_{k,l}^{(m)}|} \right\}, \qquad (5)$$

with $\lambda = 2\sqrt{\lambda_1 \lambda_2}$, i.e., $\hat{\theta}_{k,l}^{(m)} = \tilde{\theta}_{k,l}^{(m)}$.

The proof can be found in the supplementary document. It can also be found in the supplementary material how this formulation will serve the purpose of the composite parameterization, i.e., common information and specific information can be estimated separately and simultaneously.

The optimization problem (5) is a non-convex optimization problem that is difficult to solve. We address this problem by using an iterative two-stage procedure known as Difference of Convex functions (DC) programming [39]. A full description of the algorithm can be found in the supplemental material.

# 4    Simulation studies

In this section, we conduct experiments to compare the performance of the proposed SCLDA with sparse LDA (SLDA) [42] and multitask feature selection [31]. Specifically, as we focus on LDA, we use the multitask feature selection method developed in [31] on LDA, denoted as MSLDA. Both SLDA and MSLDA adopt convex regularizations. Specifically, SLDA selects features from one single data source with L1-regularization; MSLDA selects features from multiple data sources with L2/L1 regularization.

We evaluate the performances of these three methods across various parameters settings, including the number of variables, $p$, the number of features, $l$, the number of data sources, M, sample size, $n$, and the degree of overlapping of the features across different data sources, s% (the larger the $s\%$, the more shared features among the datasets). Definition of $s\%$ can be found in the simulation procedure that is included in the supplemental material. For each specification of the parameters settings, $M$ datasets can be generated following the simulation procedure. We apply the proposed SCLDA to the $M$ datasets, and identify one feature vector $\hat{\theta}^{(i)}$ for each dataset, with $\lambda$ and $q$ chosen by the method described in section 3.3. The result can be described by the number of true positives (TPs) as well as the number of false positives (FPs). Here, true positives are the non-zero elements in the learned feature vector $\hat{\theta}^{(i)}$ which are also non-zero in the $\beta_i$; false positives are the non-zero elements in $\hat{\theta}^{(i)}$, which are actually zero in $\beta_i$. As there are $m$ pairs of the TPs and FPs for the $M$ datasets, the average TP over the M datasets and the average FP over the M datasets are used as the performance measures. This procedure (i.e., from data simulation, to SCLDA, to TPs and FPs generation) can be repeated for $B$ times, and $B$ pairs of average TP and average FP are collected for SCLDA. In a similar way, we can obtain $B$ pairs of average TP and average FP for both SLDA and MSLDA.

Figures 1 (a) and (b) show comparison between SCLDA, SLDA and MSLDA by scattering the average TP against the average FP for each method. Each point corresponds to one of the N repetitions. The comparison is across various parameters settings, including the number of variables ($p = 100,200,500$), the number of data sources ($m = 2,5,10$), and the degree of overlapping of the features across different data sources ($s\% = 90\%, 70\%$). Additionally, $n/p$ is kept constant, $n/p = 1$. A general observation is that SCLDA is better than SLDA and MSLDA across all the parameter settings. Some specific trends can be summarized as follows: (i) Both SCLDA and MSLDA outperform SLDA in terms of TPs; SCLDA further outperforms MSLDA in terms of FPs. (ii) In Figure 2 (a), rows correspond to different numbers of data sources, i.e., $m = 2,5,10$ respectively. It is clear that the advantage of SCLDA over both SLDA and MSLDA is more significant when there are more data sources. Also, MSLDA performs consistently better than SLDA. Similar phenomena are shown in Figure 2 (b). This demonstrates that in analyzing each data source, both SCLDA and MSLDA are able to make use of the information contained in other data sources. SCLDA can use this information more efficiently, as SCLDA can produce less shrunken parameter estimates than MSLDA and thus it is able to preserve weak-effect features. (iii) Comparing Figures 2 (a) and (b), it can be seen that the advantage of SCLDA or MSLDA over SLDA is more significant as the data sources have more degree of overlapping in their

features. Finally, although not presented here, our simulation shows that the three methods perform similarly when $s\% = 40$ or less.

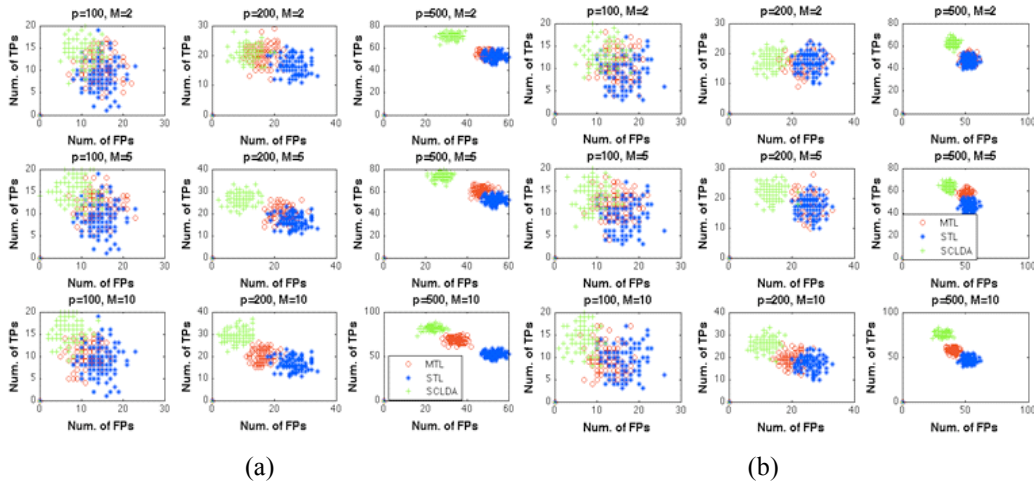

(a)                                                                    (b)

Figure 1: Average numbers of TPs vs FPs for SCLDA (green symbols "+"), SLDA (blue symbols "*") and MSLDA (red symbols "o") (a) $s\% = 90\%, n/p = 1$; (b) $s\% = 70\%, n/p = 1$

## 5      Case study

### 5.1      Data preprocessing

Our study includes 49 AD patient and 67 age-matched normal controls (NC), with each subject of AD or NC being scanned both by PET and MRI. The PET and MRI images can be downloaded from the database by the Alzheimer's Disease Neuroimaging Initiative. In what follows, we outline the data preprocessing steps.

Each image is spatially normalized to the Montreal Neurological Institute (MNI) template, using the affine transformation and subsequent non-linear wraping algorithm [43] implemented in the SPM MATLAB toolbox. This is to ensure that each voxel is located in the same anatomical region for all subjects, so that spatial locations can be reported and interpreted in a consistent manner. Once all the images in the MNI template, we further apply the Automated Anatomical Labeling (AAL) technique [43] to segment the whole brain of each subject into 116 brain regions. The 90 regions that belong to the cerebral cortex are selected for the later analysis, as the other regions are not included in the cerebral cortex are rarely considered related with AD in the literature.   The measurement of each region in the PET data is regional cerebral blood flow (rCBF); the measurement of each region in the MRI data is the structural volume of the region.

### 5.2      Disease-related brain regions

SCLDA is applied to the preprocessed PET and MRI data of AD and NC with the penalty parameter selected by the AIC method mentioned in section 3. 26 disease-related brain regions are identified from PET and 21 from MRI (see Table 1 for their names). The maps of the disease-related brain regions identified from PET and MRI are highlighted in Figure 2 (a) and (b), respectively, with different colors given to neighboring regions in order to distinguish them. Each figure is a set of horizontal cut away slices of the brain as seen from the top, which aims to provide a full view of locations of the regions.

One major observation is that the identified disease-related brain regions from MRI are in the hippocampus, parahippocampus, temporal lobe, frontal lobe, and precuneus, which is consistent with the existing literature that reports structural atrophy in these brain areas. [3-6,12-14]. The identified disease-related brain regions from PET are in the temporal, frontal and parietal lobes, which is consistent with many functional neuroimaging studies that report reduced rCBF or

reduced cortical glucose metabolism in these areas [8-10, 12-14]. Many of these identified disease-related regions can be explained in terms of the AD pathology. For example, hippocampus is a region affected by AD the earliest and severely [6] Also, as regions in the temporal lobe are essential for memory, damage on these regions by AD can explain the memory loss which is a major clinic symptom of AD. The consistency of our findings with the AD literature supports effectiveness of the proposed SCLDA.

Another finding is that there is a large overlap between the identified disease-related regions from PET and those from MRI, which implies strong interaction between functional and structural alterations in these regions. Although well-accepted biological mechanisms underlying this interaction are still not very clear, there are several explanations existing in the literature. The first explanation is that both functional and structural alterations could be the consequence of dendritic arborizations, which results from intracellular accumulation of PHFtau and further leads to neuron death and grey matter loss [14]. The second explanation is that the AD pathology may include a vascular component, which may result in reduced rCBF due to limited blood supply and may ultimately result in structural alteration such as brain atrophy [45].

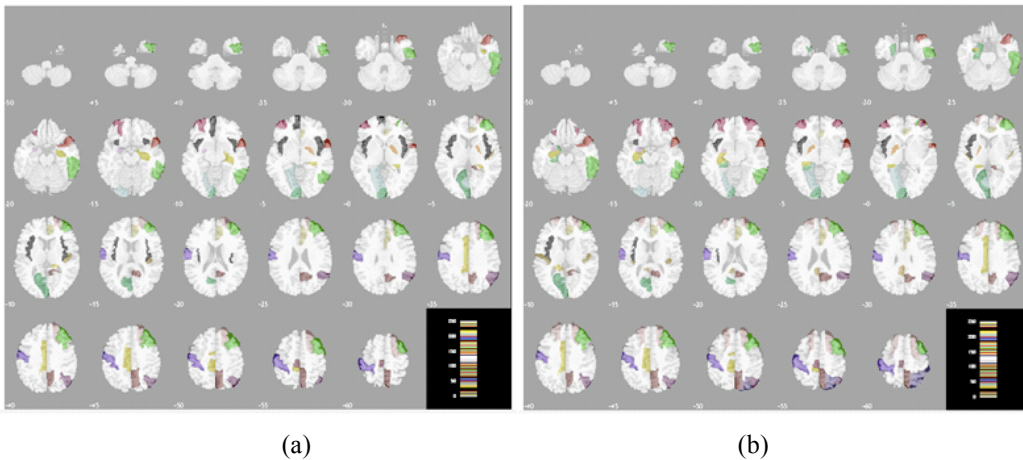

(a)                                                    (b)

Figure 2: locations of disease-related brain regions identified from (a) MRI; (b) PET

### 5.3    Classification accuracy

As one of our primary goals is to distinguish AD from NC, the identified disease-related brain regions through SCLDA are further utilized for establishing a classification model.  Specifically, for each subject, the rCBF values of the 26 disease-related brain regions identified from PET and the structural volumes of the 21 disease-related brain regions identified from MRI are used, as a joint spatial pattern of both brain physiology and structure. As a result, each subject is associated with a vector with 47 features/variables. Linear SVM (Support Vector Machine) is employed as the classifier. The classification accuracy based on 10-fold cross-validation is 94.3%. For comparison purposes, MSLDA is also applied, which identifies 45 and 38 disease-related brain regions for PET and MRI, respectively. Linear SVM applied to the 45+38 features gives a classification accuracy of only 85.8%. Note that MSLDA identifies a much larger number of disease-related brain regions than SCLDA, but some of the identified regions by MSLDA may indeed be disease-irrelevant, so including them deteriorates the classification.

### 5.4    Relationship between structural atrophy and abnormal rCBF, and severity of cognitive impairment in AD

In addition to classification, it is also of interest to further verify relevance of the identified disease-related regions with AD in an alternative way. One approach is to investigate the degree to which those disease-related regions are relevant to cognitive impairment that can be measured by the Alzheimer's disease assessment scale – cognitive subscale (ADAS-cog). ADAS measures severity of the most important symptoms of AD, while its subscale, ADAS-cog, is the most

popular cognitive testing instrument used in clinic trails. The ADAS-cog consists of 11 items measuring disturbances of memory, language, praxis, attention and other cognitive abilities that are often affected by AD. As the total score of these 11 items provides an overall assessment of cognitive impairment, we regress this ADAS-cog total score (the response) against the rCBF or structure volume measurement (the predictor) of each identified brain region, using a simple regression. The regression results are listed in Table 1.

It is not surprising to find that some regions in the hippocampus area and temporal lobes are among the best predictors, as these regions are extensively reported in the literature as the most severely affected by AD [3-6]. Also, it is found that most of these brain regions are weak-effect predictors, as most of them can only explain a small portion of the variability in the ADAS-cog total score, i.e., many R-square values in Table 1 are less than 10%. However, although the effects are weak, most of them are significant, i.e., most of the p-values in Table 1 are smaller than 0.05. Furthermore, it is worth noting that 70.22% variability in ADAS-cog can be explained by taking all the 26 brain regions identified from PET as predictors in a multiple regression model; 49.72% variability can be explained by taking all the 21 brain regions from MRI as predictors in a multiple regression model. All this findings imply that the disease-related brain regions are indeed weak-effect features if considered individually, but jointly they can play a strong role for characterizing AD. This verifies the suitability of the proposed SCLDA for AD studies, as SCLDA can preserve weak-effect features.

Table 1: Explanatory power of regional rCBF and structural volume for variability in ADAS-cog ("~" means this region is not identified from PET (or MRI) as a disease-related region by SCLDA)

| Brain regions | PET | | MRI | | Brain regions | PET | | MRI | |
|---|---|---|---|---|---|---|---|---|---|
| | $R^2$ | p-value | $R^2$ | p-value | | $R^2$ | p-value | $R^2$ | p-value |
| Precentral_L | 0.003 | 0.503 | 0.027 | 0.077 | Amygdala_L | 0.090 | 0.001 | 0.313 | $<10^{-4}$ |
| Precentral_R | 0.044 | 0.022 | ~ | ~ | Calcarine_L | 0.038 | 0.034 | 0.028 | 0.070 |
| Frontal_Sup_L | 0.051 | 0.013 | 0.047 | 0.018 | Lingual_L | 0.066 | 0.005 | 0.044 | 0.023 |
| Frontal_Sup_R | 0.044 | 0.023 | ~ | ~ | Postcentral_L | 0.038 | 0.035 | 0.026 | 0.081 |
| Frontal_Mid_R | 0.056 | 0.010 | 0.072 | 0.003 | Parietal_Sup_R | 0.001 | 0.677 | ~ | ~ |
| Frontal_M_O_L | 0.036 | 0.040 | 0.086 | 0.001 | Angular_R | 0.173 | $<10^{-4}$ | 0.063 | 0.006 |
| Frontal_M_O_R | 0.019 | 0.138 | 0.126 | 0.000 | Precuneus_R | 0.063 | 0.006 | 0.025 | 0.084 |
| Insula_L | 0.016 | 0.171 | 0.163 | $<10^{-4}$ | Paracentr_Lobu_L | 0.035 | 0.043 | 0.000 | 0.769 |
| Insula_R | ~ | ~ | 0.125 | 0.000 | Pallidum_L | 0.082 | 0.001 | ~ | ~ |
| Cingulum_A_R | 0.004 | 0.497 | 0.082 | 0.001 | Pallidum_R | ~ | ~ | 0.020 | 0.122 |
| Cingulum_Mid_L | 0.001 | 0.733 | 0.040 | 0.030 | Heschl_L | 0.001 | 0.640 | ~ | ~ |
| Cingulum_Post_L | 0.184 | $<10^{-4}$ | ~ | ~ | Heschl_R | 0.000 | 0.744 | 0.111 | 0.000 |
| Hippocampus_L | 0.158 | $<10^{-4}$ | ~ | ~ | Temporal_P_S_R | 0.008 | 0.336 | 0.071 | 0.003 |
| Hippocampus_R | ~ | ~ | 0.242 | $<10^{-4}$ | Temporal_Inf_R | 0.187 | $<10^{-4}$ | 0.147 | $<10^{-4}$ |
| ParaHippocamp_L | 0.206 | $<10^{-4}$ | ~ | ~ | All regions | 0.702 | $<10^{-4}$ | 0.497 | $<10^{-4}$ |

# 6    Conclusion

In the paper, we proposed a SCLDA model for identification of disease-related brain regions of AD from multi-modality data, which is capable to preserve weak-effect disease-related brain regions due to its less shrinkage imposed on its parameters. We applied SCLDA to the PET and MRI data of early AD patients and normal controls. As MRI and PET measure two complementary aspects (structural and functional aspects, respectively) of the same AD pathology, fusion of these two image modalities can make effective use of their interaction and thus improve the statistical power in identification of disease-related brain regions. Our findings were consistent with the literature and also showed some new aspects that may suggest further investigation in neuroimaging research in the future.

# References

[1]   deToledo-Morrell, L., Stoub, T.R., Bulgakova, M. 2004. MRI-derived entorhinal volume is a good predictor of conversion from MCI to AD. *Neurobiol. Aging* 25, 1197–1203.

[2]   Morra, J.H., Tu, Z. Validation of automated hippocampal segmentation method. NeuroImage 43, 59–68, 2008.

[3]   Morra, J.H., Tu, Z. 2009a. Automated 3D mapping of hippocampal atrophy. *Hum. Brain Map*. 30, 2766–2788.

[4]   Morra, J.H., Tu, Z. 2009b. Automated mapping of hippocampal atrophy in 1-year repeat MRI data. *NeuroImage* 45, 213-221.

[5]   Schroeter, M.L., Stein, T. 2009. Neural correlates of AD and MCI. *NeuroImage* 47, 1196–1206.

[6]   Braak, H., Braak, E. 1991. Neuropathological stageing of Alzheimer-related changes. *Acta Neuro*. 82, 239–259.

[7]   Bradley, K.M., O'Sullivan. 2002. Cerebral perfusion SPET correlated with Braak pathological stage in AD. *Brain* 125, 1772–1781.

[8]   Keilp, J.G., Alexander, G.E. 1996. Inferior parietal perfusion, lateralization, and neuropsychological dysfunction in AD. *Brain Cogn*. 32, 365–383.

[9]   Schroeter, M.L., Stein, T. 2009. Neural correlates of AD and MCI. *NeuroImage* 47, 1196–1206.

[10]  Asllani, I., Habeck, C. 2008. Multivariate and univariate analysis of continuous arterial spin labeling perfusion MRI in AD. J. *Cereb. Blood Flow Metab*. 28, 725–736.

[11]  Du,A.T., Jahng, G.H. 2006. Hypoperfusion in frontotemporal dementia and AD. *Neurology* 67, 1215–1220.

[12]  Ishii, K., Kitagaki, H. 1996. Decreased medial temporal oxygen metabolism in AD. *J. Nucl. Med*. 37, 1159–1165.

[13]  Johnson, N.A., Jahng, G.H. 2005. Pattern of cerebral hypoperfusion in AD. *Radiology* 234, 851–859.

[14]  Wolf, H., Jelic, V. 2003. A critical discussion of the role of neuroimaging in MCI. *Acta Neuroal*: 107 (4), 52-76.

[15]  Tosun, D., Mojabi, P. 2010. Joint analysis of structural and perfusion MRI for cognitive assessment and classification of AD and normal aging. *NeuroImage* 52, 186-197.

[16]  Alsop, D., Casement, M. 2008. Hippocampal hyperperfusion in Alzheimer's disease. *NeuroImage* 42, 1267–1274.

[17]  Mosconi, L., Tsui, W.-H. 2005. Reduced hippocampal metabolism in MCI and AD. *Neurology* 64, 1860–1867.

[18]  Mulert, C., Lemieux, L. 2010. *EEG-fMRI: physiological basis, technique and applications*. Springer.

[19]  Xu, L., Qiu, C., Xu, P. and Yao, D. 2010. A parallel framework for simultaneous EEG/fMRI analysis: methodology and simulation. *NeuroImage*, 52(3), 1123-1134.

[20]  Philiastides, M. and Sajda, P. 2007. EEG-informed fMRI reveals spatiotemporal characteristics of perceptual decision making. *Journal of Neuroscience*, 27(48), 13082-13091.

[21]  Daunizeau, J., Grova, C. 2007. Symmetrical event-related EEG/fMRI information fusion. *NeuroImage* 36, 69-87.

[22]  Jagust, W. 2006. PET and MRI in the diagnosis and prediction of dementia. *Alzheimer's Dement* 2, 36-42.

[23]  Kawachi, T., Ishii, K. and Sakamoto, S. 2006. Comparison of the diagnostic performance of FDG-PET and VBM. *Eur.J.Nucl.Med.Mol.Imaging* 33, 801-809.

[24]  Matsunari, I., Samuraki, M. 2007. Comparison of 18F-FDG PET and optimized voxel-based morphometry for detection of AD. *J.Nucl.Med* 48, 1961-1970.

[25]  Schmidt, M., Fung, G. and Rosales, R. 2007. Fast optimization methods for L1-regularization: a comparative study and 2 new approaches. *ECML* 2007.

[26]  Liu, J., Ji, S. and Ye, J. 2009. *SLEP: sparse learning with efficient projections,* Arizona state university.

[27]  Tibshirani, R. 1996. Regression Shrinkage and Selection via the Lasso, *JRSS, Series B*, 58(1):267–288.

[28]  Friedman, J., Hastie, T. and Tibshirani, R. 2007. Sparse inverse covariance estimation with the graphical lasso. *Biostatistics*, 8(1):1–10.

[29]  Zou, H., Hastie, T. and Tibshirani, R. 2006. Sparse PCA*, J. of Comp. and Graphical Statistics*, 15(2), 262-286.

[30]  Qiao, Z., Zhou, L and Huang, J. 2006. Sparse LDA with applications to high dimensional low sample size data. *IAENG applied mathematics*, 39(1).

[31]  Argyriou, A., Evgeniou, T. and Pontil, M. 2008. Convex multi-task feature learning. Machine Learning 73(3): 243–272.

[32]  Huang, S., Li, J., et al. 2010. Learning Brain Connectivity of AD by Sparse Inverse Covariance Estimation, *NeuroImage*, 50, 935-949.

[33]  Candes, E., Wakin, M. and Boyd, S. 2008. Enhancing sparsity by reweighted L1 minimization. *Journal of Fourier analysis and applications*, 14(5), 877-905.

[34]   Mazumder, R.; Friedman, J. 2009. SparseNet: Coordinate Descent with Non-Convex Penalties.  Manuscript.

[35]  Zhang, T. 2008. Multi-stage Convex Relaxation for Learning with Sparse Regularization. *NIPS* 2008.

[36]  Campbell, N. 1984. Canonical variate analysis ageneral formulation. *Australian Jour of Stat* 26, 86–96.

[37]  Hastie, T. and Tibshirani, R. 1994. *Discriminant analysis by gaussian mixtures*. Technical report. AT&T Bell Lab.

[38]  Kumar, N. and Andreou, G. 1998. Heteroscedastic discriminant analysis and reduced rank HMMs for improved speech recognition. *Speech Communication*, 26 (4), 283-297.

[39]  Gasso, G., Rakotomamonjy, A. and Canu, S. 2009. Recovering sparse signals with non-convex penalties and DC programming. *IEEE Trans. Signal Processing* 57( 12), 4686-4698.

[40]  Guo, J., Levina, E., Michailidis, G. and Zhu, J. 2011. Joint estimation of multiple graphical models. *Biometrika* 98(1) 1-15.

[41]  Bertsekas, D. 1982. Projected newton methods for optimization problems with simple constraints. *SIAM J. Control Optim* 20, 221-246.

[42]  Clemmensen, L., Hastie, T., Witten, D. and Ersboll:, B. 2011. Sparse Discriminant Analysis. *Technometrics* (in press)

[43]  Friston, K.J., Ashburner, J. 1995. Spatial registration and normalization of images. *HBM* 2, 89–165.

[44]  Tzourio-Mazoyer, N., et al., 2002. Automated anatomical labelling of activations in SPM. *NeuroImage* 15, 273–289.

[45]  Bidzan, L. 2005. Vascular factors in dementia. *Psychiatr. Pol*. 39, 977-986.

